# Beyond Gaussian Processes: On the Distributions of Infinite Networks

**Ricky Der**
Department of Mathematics
University of Pennsylvania
Philadelphia, PA 19104
rickyder@math.upenn.edu

**Daniel Lee**
Department of Electrical Engineering
University of Pennsylvania
Philadelphia, PA 19104
ddlee@seas.upenn.edu

## Abstract

A general analysis of the limiting distribution of neural network functions is performed, with emphasis on non-Gaussian limits. We show that with i.i.d. symmetric stable output weights, and more generally with weights distributed from the normal domain of attraction of a stable variable, that the neural functions converge in distribution to stable processes. Conditions are also investigated under which Gaussian limits do occur when the weights are independent but not identically distributed. Some particularly tractable classes of stable distributions are examined, and the possibility of learning with such processes.

## 1  Introduction

Consider the model

$$f_n(x) = \frac{1}{s_n} \sum_{j=1}^{n} v_j h(x; u_j) \equiv \frac{1}{s_n} \sum_{j=1}^{n} v_j h_j(x) \tag{1}$$

which can be viewed as a multi-layer perceptron with input $x$, hidden functions $h$, weights $u_j$, output weights $v_j$, and $s_n$ a sequence of normalizing constants. The work of Radford Neal [1] showed that, under certain assumptions on the parameter priors $\{v_j, h_j\}$, the distribution over the implied network functions $f_n$ converged to that of a Gaussian process, in the large network limit $n \to \infty$. The main feature of this derivation consisted of an invocation of the classical Central Limit Theorem (CLT).

While one cavalierly speaks of "the" central limit theorem, there are in actuality many different CLTs, of varying generality and effect. All are concerned with the limits of suitably normalised sums of independent random variables (or where some condition is imposed so that no one variable dominates the sum[1]), but the limits themselves differ greatly: Gaussian, stable, infinitely divisible, or, discarding the infinitesimal assumption, none of these. It follows that in general, the asymptotic process for (1) may not be Gaussian. The following questions then arise: what is the relationship between choices of distributions on the model priors, and the asymptotic distribution over the induced neural functions? Under what conditions does the Gaussian approximation hold? If there do exist non-Gaussian limit points, is it possible to construct analagous generalizations of Gaussian process regression?

Previous work on these problems consists mainly in Neal's publication [1], which established that when the output weights $v_j$ are *finite variance* and *i.i.d.*, the limiting distribution is a Gaussian process. Additionally, it was shown that when the weights are i.i.d. symmetric stable (SS), the first-order marginal distributions of the functions are also SS. Unfortunately, no mathematical analysis was presented to show that the higher-order distributions converged, though empirical evidence was suggestive of that hypothesis. Moreover, the exact form of the higher-dimensional distributions remained elusive.

This paper conducts a further investigation of these questions, with concentration on the cases where the weight priors can be 1) of infinite variance, and 2) non-i.i.d. Such assumptions fall outside the ambit of the classical CLT, but are amenable to more general limit methods. In Section 1, we give a general classification of the possible limiting processes that may arise under an i.i.d. assumption on output weights distributed from a certain class — roughly speaking, those weights with tails asymptotic to a power-law — and provide explicit formulae for all the joint distribution functions. As a byproduct, Neal's preliminary analysis is completed, a full multivariate prescription attained and the convergence of the finite-dimensional distributions proved. The subsequent section considers non-i.i.d. priors, specifically independent priors where the "identically distributed" assumption is discarded. An example where a finite-variance non-Gaussian process acts as a limit point for a non-trivial infinite network is presented, followed by an investigation of conditions under which the Gaussian approximation is valid, via the Lindeberg-Feller theorem. Finally, we raise the possibility of replacing network models with the processes themselves for learning applications: here, motivated by the foregoing limit theorems, the set of stable processes form a natural generalization to the Gaussian case. Classes of stable stochastic processes are examined where the parameterizations are particularly simple, as well as preliminary applications to the nonlinear regression problem.

## 2 Neural Network Limits

Referring to (1), we make the following assumptions: $h_j(x) \equiv h(x; u_j)$ are uniformly bounded in $x$ (as for instance occurs if $h$ is associated with some fixed nonlinearity), and $\{u_j\}$ is an i.i.d. sequence, so that $h_j(x)$ are i.i.d. for fixed $x$, and independent of $\{v_j\}$. With these assumptions, the choice of output priors $v_j$ will tend to dictate large-network behavior, independently of $u_j$. In the sequel, we restrict ourselves to functions $f_n(x) : \mathbb{R} \to \mathbb{R}$, as the respective proofs for the generalizations of $x$ and $f_n$ to higher-dimensional spaces are routine. Finally, all random variables are assumed to be of zero mean whenever first moments exist. For brevity, we only present sketches of proofs.

### 2.1 Limits with i.i.d. priors

The Gaussian distribution has the feature that if $X_1$ and $X_2$ are statistically independent copies of the Gaussian variable $X$, then their linear combination is also Gaussian, i.e. $aX_1 + bX_2$ has the same distribution as $cX + d$ for some $c$ and $d$. More generally, the *stable* distributions [5], [6, Chap. 17] are defined to be the set of all distributions satisfying the above "closure" property. If one further demands symmetry of the distribution, then they must have characteristic function $\Phi(t) = e^{-\sigma^\alpha |t|^\alpha}$, for parameters $\sigma > 0$ (called the spread), and $0 < \alpha \leq 2$, termed the index. Since the characteristic functions are not generally twice differentiable at $t = 0$, their variances are infinite, the Gaussian distribution being the only finite variance stable distribution, associated to index $\alpha = 2$.

The attractive feature of stable variables, by definition, is closure under the formation of linear combinations: the linear combination of any two independent stable variables is another stable variable of the same index. Moreover, the stable distributions are attraction points of distributions under a linear combiner operator, and indeed, the only such distributions in

the following sense: if $\{Y_j\}$ are i.i.d., and $a_n + \frac{1}{s_n} \sum_{j=1}^{n} Y_j$ converges in distribution to $X$, then $X$ must be stable [5]. This fact already has consequences for our network model (1), and implies that — under i.i.d. priors $v_j$, and assuming (1) converges at all — convergence can occur only to stable variables, for each $x$.

Multivariate analogues are defined similarly: we say a random vector $\mathbf{X}$ is (strictly) stable if, for every $a, b \in \mathbb{R}$, there exists a constant $c$ such that $a\mathbf{X}_1 + b\mathbf{X}_2 = c\mathbf{X}$ where $\mathbf{X}_i$ are independent copies of $\mathbf{X}$ and the equality is in distribution. A *symmetric* stable random vector is one which is stable and for which the distribution of $\mathbf{X}$ is the same as $-\mathbf{X}$. The following important classification theorem gives an explicit Fourier domain description of all multivariate symmetric stable distributions:

**Theorem 1.** Kuelbs [5]. $\mathbf{X}$ *is a symmetric $\alpha$-stable vector if and only if it has characteristic function*

$$\Phi(\mathbf{t}) = \exp\left\{ - \int_{S^{d-1}} |\langle \mathbf{t}, \mathbf{s} \rangle|^{\alpha} \, d\Gamma(\mathbf{s}) \right\} \tag{2}$$

*where $\Gamma$ is a finite measure on the unit $(d-1)$-sphere $S^{d-1}$, and $0 < \alpha \leq 2$.*

*Remark:* (2) remains unchanged replacing $\Gamma$ by the symmetrized measure $\tilde{\Gamma} = \frac{1}{2}(\Gamma(A) + \Gamma(-A))$, for all Borel sets $A$. In this case, the (unique) symmetrized measure $\tilde{\Gamma}$ is called the *spectral measure* of the stable random vector $\mathbf{X}$.

Finally, stable *processes* are defined as indexed sets of random variables whose finite-dimensional distributions are (multivariate) stable.

First we establish the following preliminary result.

**Lemma 1.** *Let $v$ be a symmetric stable random variable of index $0 < \alpha \leq 2$, and spread $\sigma > 0$. Let $h$ be independent of $v$ and $E|h|^{\alpha} < \infty$. If $y = hv$, and $\{y_i\}$ are i.i.d. copies of $y$, then $S_n = \frac{1}{n^{1/\alpha}} \sum_{i=1}^{n} y_i$ converges in distribution to an $\alpha$-stable variable with characteristic function $\Phi(t) = \exp\{-|\sigma t|^{\alpha} E|h|^{\alpha}\}$.*

*Proof.* This follows by computing the characteristic function $\Phi_{S_n}$, then using standard theorems in measure theory (e.g. [4]), to obtain $\lim_{n \to \infty} \log \Phi_{S_n}(t) = -|\sigma t|^{\alpha} E|h|^{\alpha}$. ∎

Now we can state the first network convergence theorem.

**Proposition 1.** *Let the network (1) have symmetric stable i.i.d. weights $v_j$ of index $0 < \alpha \leq 2$ and spread $\sigma$. Then $f_n(x) = \frac{1}{n^{1/\alpha}} \sum_{j=1}^{n} v_j h_j(x)$ converges in distribution to a symmetric $\alpha$-stable process $f(x)$ as $n \to \infty$. The finite-dimensional stable distribution of $(f(x_1), \dots, f(x_d))$, where $x_i \in \mathbb{R}$, has characteristic function:*

$$\Psi(\mathbf{t}) = \exp\left(-\sigma^{\alpha} E_{\mathbf{h}} |\langle \mathbf{t}, \mathbf{h} \rangle|^{\alpha}\right) \tag{3}$$

*where $\mathbf{h} = (h(x_1), \dots, h(x_d))$, and $h(x)$ is a random variable with the common distribution (across $j$) of $h_j(x)$. Moreover, if $\mathbf{h} = (h(x_1), \dots, h(x_d))$ has joint probability density $p(\mathbf{h}) = p(r\mathbf{s})$, with $\mathbf{s}$ on the $S^{d-1}$ sphere and $r$ the radial component of $h$, then the finite measure $\Gamma$ corresponding to the multivariate stable distribution of $(f(x_1), \dots, f(x_d))$ is given by*

$$d\Gamma(\mathbf{s}) = \left( \int_0^{\infty} r^{\alpha+d-1} p(r\mathbf{s}) \, dr \right) d\mathbf{s} \tag{4}$$

*where $d\mathbf{s}$ is Lebesgue measure on $S^{d-1}$.*

*Proof.* It suffices to show that every finite-dimensional distribution of $f(x)$ converges to a symmetric multivariate stable characteristic function. We have $\sum_{i=1}^{d} t_i f_n(x_i) =$

$\frac{1}{n^{1/\alpha}} \sum_{j=1}^{n} v_j \sum_{i=1}^{d} t_i h_j(x_i)$ for constants $\{x_1, \ldots, x_d\}$ and $(t_1, \ldots, t_d) \in \mathbb{R}^d$. An application of Lemma 1 proves the statement. The relation between the expectation in (3) and the stable spectral measure (4) is derived from a change of variable to spherical coordinates in the $d$-dimensional space of $\mathbf{h}$. ∎

*Remark:* When $\alpha = 2$, the exponent in the characteristic function (3) is a quadratic form in $\mathbf{t}$, and becomes the usual Gaussian multivariate distribution.

The above proposition is the rigorous completion of Neal's analysis, and gives the explicit form of the asymptotic process under i.i.d. SS weights. More generally, we can consider output weights from the *normal domain of attraction of index $\alpha$*, which, roughly, consists of those densities whose tails are asymptotic to $|x|^{-(\alpha+1)}$, $0 < \alpha < 2$ [6, pg. 547]. With a similar proof to the previous theorem, one establishes

**Proposition 2.** *Let network (1) have i.i.d. weights $v_j$ from the normal domain of attraction of an SS variable with index $\alpha$, spread $\sigma$. Then $f_n(x) = \frac{1}{n^{1/\alpha}} \sum_{j=1}^{n} v_j h_j(x)$ converges in distribution to a symmetric $\alpha$-stable process $f(x)$, with the joint characteristic functions given as in Proposition 1.*

### 2.1.1 Example: Distributions with step-function priors

Let $h(x) = \text{sgn}(a + ux)$, where $a$ and $u$ are independent Gaussians with zero mean. From (3) it is clear that the limiting network function $f(x)$ is a constant (in law, hence almost surely), as $|x| \to \infty$, so that the interesting behavior occurs in some "central region" $|x| < k$. Neal in [1] has shown that when the output weights $v_j$ are Gaussian, then the choice of the signum nonlinearity for $h$ gives rise to local Brownian motion in the central regime.

There is a natural generalization of the Brownian process within the context of symmetric stable processes, called the *symmetric $\alpha$-stable Lévy motion*. It is characterised by an indexed sequence $\{w_t : t \in \mathbb{R}\}$ satisfying i) $w_0 = 0$ almost surely, ii) independent increments, and iii) $w_t - w_s$ is distributed symmetric $\alpha$-stable with spread $\sigma = |t - s|^{1/\alpha}$. As we shall now show, the choice of step-function nonlinearity for $h$ and symmetric $\alpha$-stable priors for $v_j$ lead to locally Lévy stable motion, which provide a theoretical exposition for the empirical observations in [1].

Fix two nearby positions $x$ and $y$, and select $\sigma = 1$ for notational simplicity. From (3) the random variable $f(x) - f(y)$ is symmetric stable with spread parameter $[E_{\mathbf{h}}|h(x) - h(y)|^\alpha]^{1/\alpha}$. For step inputs, $|h(x) - h(y)|$ is non-zero only when the step located at $-a/u$ falls between $x$ and $y$. For small $|x-y|$ approximate the density of this event to be uniform, so that $[E_{\mathbf{h}}|h(x) - h(y)|^\alpha] \sim |x - y|$. Hence locally, the increment $f(x) - f(y)$ is a symmetric stable variable with spread proportional to $|x - y|^{1/\alpha}$, which is condition (iii) of Lévy motion. Next let us demonstrate that the increments are independent. Consider the vector $(f(x_1)-f(x_2), f(x_2)-f(x_3), \ldots, f(x_{n-1})-f(x_n))$, where $x_1 < x_2 < \ldots < x_n$. Its joint characteristic function in the variables $t_1, \ldots, t_{n-1}$ can be calculated to be

$$\Phi(t_1, \ldots, t_{n-1}) = \exp\left(-E_{\mathbf{h}}|t_1(h(x_1) - h(x_2)) + \cdots + t_{n-1}(h(x_{n-1}) - h(x_n))|^\alpha\right) \tag{5}$$

The disjointness of the intervals $(x_{i-1}, x_i)$ implies that the only events which have non-zero probability within the range $[x_1, x_n]$ are the events $|h(x_i) - h(x_{i-1})| = 2$ for some $i$, and zero for all other indices. Letting $p_i$ denote the probabilities of those events, (5) reads

$$\Phi(t_1, \ldots, t_{n-1}) = \exp\left(-2^\alpha(p_1|t_1|^\alpha + \cdots + p_{n-1}|t_{n-1}|^\alpha)\right) \tag{6}$$

which describes a vector of independent $\alpha$-stable random variables, as the characteristic function splits. Thus the limiting process has independent increments.

The differences between sample functions arising from Cauchy priors as opposed to Gaussian priors is evident from Fig. 1, which displays sample paths from Gaussian and

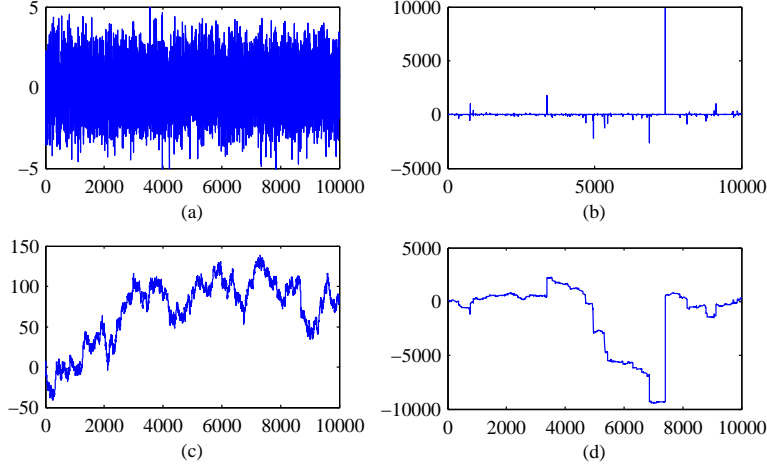

Figure 1: Sample functions: (a) i.i.d. Gaussian, (b) i.i.d. Cauchy, (c) Brownian motion, (d) Lévy Cauchy-Stable motion.

Cauchy i.i.d. processes $w_n$ and their "integrated" versions $\sum_{i=1}^{n} w_i$, simulating the Lévy motions. The sudden jumps in the Cauchy motion arise from the presence of strong outliers in the respective Cauchy i.i.d. process, which would correspond, in the network, to hidden units with heavy weighting factors $v_j$.

## 2.2 Limits with non-i.i.d. priors

We begin with an interesting example, which shows that if the "identically distributed" assumption for the output weights is dispensed with, the limiting distribution of (1) can attain a non-stable (and non-Gaussian) form. Take $v_j$ to be independent random variables with $P(v_j = 2^{-j}) = P(v_j = -2^{-j}) = 1/2$. The characteristic functions can easily be computed as $E[e^{itv_j}] = \cos(t/2^j)$. Now recall the Vieté formula:

$$\prod_{j=1}^{n} \cos(t/2^j) = \frac{\sin t}{2^n \sin(t/2^n)} \tag{7}$$

Taking $n \to \infty$ shows that the limiting characteristic function is a sinc function, which corresponds with the uniform density. Selecting the signum nonlinearity for $h$, it is not difficult to show with estimates on the tail of the product (7) that all finite-dimensional distributions of the neural process $f_n(x) = \sum_{j=1}^{n} v_j h_j(x)$ converge, so that $f_n$ converges in distribution to a random process whose first-order distributions are uniform[2].

What conditions are required on independent, but not necessarily identically distributed priors $v_j$ for convergence to the Gaussian? This question is answered by the classical Lindeberg-Feller theorem.

**Theorem 2.** Central Limit Theorem (Lindeberg-Feller) [4]. *Let $v_j$ be a sequence of independent random variables each with zero mean and finite variance, define $s_n^2 = var[\sum_{j=1}^{n} v_j]$, and assume $s_1 \neq 0$. Then the sequence $\frac{1}{s_n} \sum_{j=1}^{n} v_j$ converges in distri-*

*bution to an $N(0,1)$ variable, if*

$$\lim_{n\to\infty} \frac{1}{s_n^2} \sum_{i=1}^{n} \int_{|v|\geq\epsilon s_n} v^2 \, dF_{v_j}(v) = 0 \qquad (8)$$

*for each $\epsilon > 0$, and where $F_{v_j}$ is the distribution function for $v_j$.*

Condition (8) is called the Lindeberg condition, and imposes an "infinitesimal" requirement on the sequence $\{v_j\}$ in the sense that no one variable is allowed to dominate the sum. This theorem can be used to establish the following non-i.i.d. network convergence result.

**Proposition 3.** *Let the network (1) have independent finite-variance weights $v_j$. Defining $s_n^2 = var[\sum_{j=1}^{n} v_j]$, if the sequence $\{v_j\}$ is Lindeberg then $f_n(x) = \frac{1}{s_n} \sum_{j=1}^{n} v_j h_j(x)$ converges in distribution to a Gaussian process $f(x)$ of mean zero and covariance function $C(f(x), f(y)) = E[h(x)h(y)]$ as $n \to \infty$, where $h(x)$ is a variable with the common distribution of the $h_j(x)$.*

*Proof.* Fix a finite set of points $\{x_1, \ldots, x_k\}$ in the input space, and look at the joint distribution $(f_n(x_1), \ldots, f_n(x_n))$. We want to show these variables are jointly Gaussian in the limit as $n \to \infty$, by showing that every linear combination of the components converges in distribution to a Gaussian distribution. Fixing $k$ constants $\mu_i$, we have $\sum_{i=1}^{k} \mu_i f(x_i) = \frac{1}{s_n} \sum_{j=1}^{n} v_j \sum_{i=1}^{k} \mu_i h_j(x_i)$. Define $\xi_j = \sum_{i=1}^{k} \mu_i h_j(x_i)$, and $\tilde{s}_n^2 = var(\sum_{j=1}^{n} v_j \xi_j) = (E\xi^2) s_n^2$, where $\xi$ is a random variable with the common distribution of $\xi_j$. Then for some $c > 0$:

$$\frac{1}{\tilde{s}_n^2} \sum_{j=1}^{n} \int_{|v_j \xi_j| \geq \epsilon \tilde{s}_n} |v_j(\omega) \xi_j(\omega)|^2 \, dP(\omega) \leq \frac{c^2}{E\xi^2} \frac{1}{s_n^2} \sum_{j=1}^{n} \int_{|v_j| \geq \epsilon \frac{(E\xi^2)^{1/2} s_n}{c}} |v_j(\omega)|^2 \, dP(\omega)$$

The right-hand side can be made arbitrarily small, from the Lindeberg assumption on $\{v_j\}$, hence $\{v_j \xi_j\}$ is Lindeberg, from which the theorem follows. The covariance function is easy to calculate. ∎

**Corollary 1.** *If the output weights $\{v_j\}$ are a uniformly bounded sequence of independent random variables, and $\lim_{n\to\infty} s_n = \infty$, then $f_n(x)$ in (1) converges in distribution to a Gaussian process.*

The preceding corollary, besides giving an easily verifiable condition for Gaussian limits, demonstrates that the non-Gaussian convergence in the example initialising Section 2.2 was made possible precisely because the weights $v_j$ decayed sufficiently quickly with $j$, with the result that $\lim_n s_n < \infty$.

## 3 Learning with Stable Processes

One of the original reasons for focusing machine learning interest on Gaussian processes consisted in the fact that they act as limit points of suitably constructed parametric models [2], [3]. The problem of learning a regression function, which was previously tackled by Bayesian inference on a modelling neural network, could be reconsidered by directly placing a Gaussian process prior on the fitting functions themselves. Yet already in early papers introducing the technique, reservations had been expressed concerning such wholesale replacement [2]. Gaussian processes did not seem to capture the richness of finite neural networks — for one, the dependencies between multiple outputs of a network vanished in the Gaussian limit.

Consider the simplest regression problem, that of the estimation of a state process $u(x)$ from observations $y(x_i)$, under the model

$$y(x) = u(x) + \epsilon(x) \qquad (9)$$

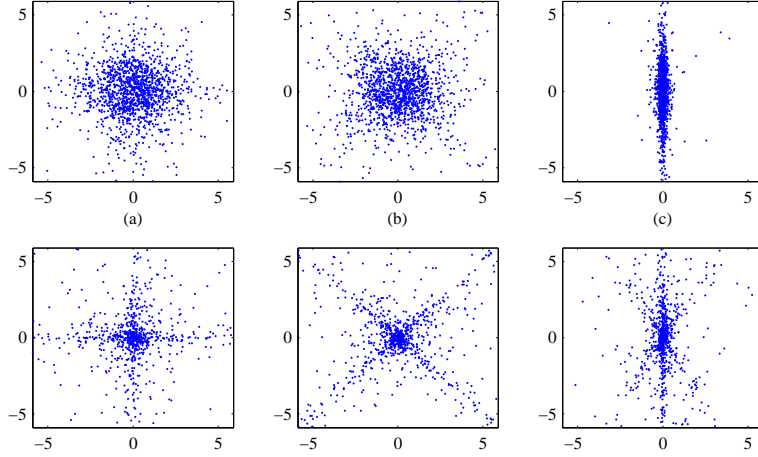

Figure 2: Scatter plots of bivariate symmetric $\alpha$-stable distributions with discrete spectral measures. Top row: $\alpha = 1.5$; Bottom row: $\alpha = 0.5$. Left to right: (a) $\mathbf{H}$ = identity, (b) $\mathbf{H}$ a rotation, (c) $\mathbf{H}$ a $2 \times 3$ matrix with columns $(-1/16, \sqrt{3}/16)^T$, $(0, 1)^T$, $(1/16, \sqrt{3}/16)^T$.

where $\epsilon(x)$ is noise independent of $u$. The obvious generalization of Gaussian process regression involves the placement of a stable process prior of index $\alpha$ on $u$, and setting $\epsilon$ as i.i.d. stable noise of the same index. Then the observations $y$ also form a stable process of index $\alpha$. Two advantages come with such generalization. First, the use of a heavy-tailed distribution for $\epsilon$ will tend to produce more robust regression estimates, relative to the Gaussian case; this robustness can be additionally controlled by the stability parameter $\alpha$. Secondly, a glance at the classification of Theorem 1 indicates that the correlation structure of stable vectors (hence processes), is significantly richer than that of the Gaussian; the space of $n$-dimensional stable vectors is already characterised by a whole space of measures, rather than an $n \times n$ covariance matrix. The use of such priors on the data $u$ afford a significant broadening in the number of interesting dependency relationships that may be assumed.

An understanding of the dependency structure of multivariate stable vectors can be first broached by considering the following basic class. Let $\mathbf{v}$ be a vector of i.i.d. symmetric stable variables of the same index, and let $\mathbf{H}$ be a matrix of appropriate dimension so that $\mathbf{x} = \mathbf{Hv}$ is well-defined. Then $\mathbf{x}$ has a symmetric stable characteristic function, where the spectral measure $\tilde{\Gamma}$ in Theorem 1 is *discrete*, i.e. concentrated on a finite number of points. Divergences in the correlation structure are readily apparent even within this class. In the Gaussian case, there is no advantage in the selection of non-square matrices $\mathbf{H}$, since the distribution of $\mathbf{x}$ can always be obtained by a square mixing matrix $\tilde{\mathbf{H}}$ with the same number of rows as $\mathbf{H}$. Not so when $\alpha < 2$, for then the characteristic function for $\mathbf{x}$ in general possesses $n$ fundamental discontinuities in higher-order derivatives, where $n$ is the number of columns of $\mathbf{H}$. Furthermore, in the square case, replacement of $\mathbf{H}$ with $\mathbf{HR}$, where $\mathbf{R}$ is any rotation matrix, leaves the distribution invariant when $\alpha = 2$; for non-Gaussian stable vectors, the mixing matrices $\mathbf{H}$ and $\mathbf{H}'$ give rise to the same distribution only when $|\mathbf{H}^{-1}\mathbf{H}'|$ is a permutation matrix, where $|\cdot|$ is defined component-wise. Figure 2 illustrates the variety of dependency structures which can be attained as $\mathbf{H}$ is changed. A number of techniques already exist in the statistical literature for the estimation of the spectral measure (and hence the mixing $\mathbf{H}$) of multivariate stable vectors from empirical data. The infinite-dimensional generalization of the above situation gives rise to the set of stable processes produced as time-varying filtered versions of i.i.d. stable noise, and

similar to the Gaussian process, are parameterized by a centering (mean) function $\mu(x)$ and a bivariate filter function $h(x, \nu)$ encoding dependency information. Another simple family of stable processes consist of the so-called *sub-Gaussian processes*. These are processes defined by $u(x) = A^{1/2}G(x)$ where $A$ is a totally right-skew $\alpha/2$ stable variable [5], and $G$ a Gaussian process of mean zero and covariance $K$. The result is a symmetric $\alpha$-stable random process with finite-dimensional characteristic functions of form

$$\Phi(\mathbf{t}) = \exp(-\frac{1}{2}|\langle \mathbf{t}, \mathbf{Kt}\rangle|^{\alpha/2}) \tag{10}$$

The sub-Gaussian processes are then completely parameterized by the statistics of the subordinating Gaussian process $G$. Even more, they have the following *linear* regression property [5]: if $Y_1, \ldots, Y_n$ are jointly sub-Gaussian, then

$$E[Y_n|Y_1, \ldots, Y_{n-1}] = a_1 Y_1 + \cdots a_{n-1} Y_{n-1}. \tag{11}$$

Unfortunately, the regression is somewhat trivial, because a calculation shows that the coefficients of regression $\{a_i\}$ are the *same* as the case where $Y_i$ are assumed jointly Gaussian! Indeed, this curious property appears anytime the variables take the form $\mathbf{Y} = B\mathbf{G}$, for *any* fixed scalar random variable $B$ and Gaussian vector $\mathbf{G}$. It follows that the predictive mean estimates for (10) employing sub-Gaussian priors are identical to the estimates under a Gaussian hypothesis. On the other hand, the conditional *distribution* of $Y_n|Y_1, \ldots, Y_{n-1}$ differs greatly from the Gaussian, and is neither stable nor symmetric about its conditional mean in general. From Fig. 2 one even sees that the conditional distribution may be multimodal, in which case the predictive mean estimates are not particularly valuable. More useful are MAP estimates, which in the Gaussian scenario coincide with the conditional mean. In any case, regression on stable processes suggest the need to compute and investigate the entire *a posteriori* probability law.

The main thrust of our foregoing results indicate that the class of possible limit points of network functions is significantly richer than the family of Gaussian processes, even under relatively restricted (e.g. i.i.d.) hypotheses. Gaussian processes are the appropriate models of large networks with finite variance priors in which no one component dominates another, but when the finite variance assumption is discarded, stable processes become the natural limit points. Non-stable processes can be obtained with the appropriate choice of non-i.i.d. parameters priors, even in an infinite network. Our discussion of the stable process regression problem has principally been confined to an exposition of the basic theoretical issues and principles involved, rather than to algorithmic procedures. Nevertheless, since simple closed-form expressions exist for the characteristic functions, the predictive probability laws can all in principle be computed with multi-dimensional Fourier transform techniques. Stable variables form mathematically natural generalisations of the Gaussian, with some fundamental, but compelling, differences which suggest additional variety and flexibility in learning applications.

## Footnotes

[1]Typically called an *infinitesimal* condition — see [4].

[2]An intuitive proof is as follows: one thinks of $\sum_j v_j$ as a binary expansion of real numbers in [-1,1]; the prescription of the probability laws for $v_j$ imply all such expansions are equiprobable, manifesting in the uniform distribution.

## References

[1] R. Neal, *Bayesian Learning for Neural Networks*. New York: Springer-Verlag, 1996.

[2] D. MacKay. *Introduction to Gaussian Processes*. Extended lecture notes, NIPS 1997.

[3] M. Seeger, *Gaussian Processes for Machine Learning*. International Journal of Neural Systems 14(2), 2004, 69–106.

[4] C. Burrill, *Measure, Integration and Probability*. New York: McGraw-Hill, 1972.

[5] G. Samorodnitsky & M. Taqqu, *Stable Non-Gaussian Random Processes*. New York: Chapman & Hall, 1994.

[6] W. Feller, *An Introduction to Probability Theory and Its Applications, Vol. 2*. New York: John Wiley & Sons, 1966.
